# On Bootstrapping the ROC Curve

**Patrice Bertail**
CREST (INSEE) & MODAL'X - Université Paris 10
pbertail@u-paris10.fr

**Stéphan Clémençon**
Telecom Paristech (TSI) - LTCI UMR Institut Telecom/CNRS 5141
stephan.clemencon@telecom-paristech.fr

**Nicolas Vayatis**
ENS Cachan & UniverSud - CMLA UMR CNRS 8536
vayatis@cmla.ens-cachan.fr

## Abstract

This paper is devoted to thoroughly investigating how to bootstrap the ROC curve, a widely used visual tool for evaluating the accuracy of test/scoring statistics in the bipartite setup. The issue of confidence bands for the ROC curve is considered and a resampling procedure based on a smooth version of the empirical distribution called the "smoothed bootstrap" is introduced. Theoretical arguments and simulation results are presented to show that the "smoothed bootstrap" is preferable to a "naive" bootstrap in order to construct accurate confidence bands.

## 1   Introduction

Since the seminal contribution of [14], so-called ROC *curves* (ROC standing for *Receiving Operator Characteristic*) have been extensively used in a wide variety of applications (anomaly detection in signal analysis, medical diagnosis, search engines, credit-risk screening) as a visual tool for evaluating the performance of a test statistic regarding its capacity of discrimination between two populations, see [8]. Whereas the statistical properties of their empirical counterparts have been only lately studied from the asymptotic angle, see [18, 13, 11, 16], ROC curves also have recently received much attention in the machine-learning literature through the development of statistical learning procedures tailored for the *ranking problem*, see [10, 2]. The latter consists of determining, based on training data, a test statistic $s(X)$ (also called a *scoring function*) with a ROC curve "as high as possible" at all points of the ROC space. Given a candidate $s(X)$, it is thus of prime importance to assess its performance by computing a confidence band for the corresponding ROC curve, in a data-driven fashion preferably. Indeed, in such a functional setup, resampling-based procedures should naturally be preferred to those relying on computing/simulating the (gaussian) limiting distribution, as first observed in [19, 21, 20], where the use of the bootstrap is promoted for building confidence bands in the ROC space.

By building on recent works, see [17, 12], it is the purpose of this paper to investigate how the bootstrap approach should be practically implemented based on a thorough analysis of the asymptotic properties of empirical ROC curves. Beyond the pointwise analysis developed in the studies mentioned above, here we tackle the problem from a functional angle, considering the entire ROC curve or parts of it. This viewpoint indeed appears as particularly relevant in scoring applications. Although the asymptotic results established in this paper are of a theoretical nature, they are considerably meaningful from a computational perspective. It turns out indeed that *smoothing* is the

key ingredient for the bootstrap confidence band to be accurate, whereas a *naive bootstrap* approach would yield bands of low coverage probability in this case and should be consequently avoided by practicioners for analyzing ROC curves.

The rest of the paper is organized as follows. In Section 2, notations are first set out and certain key notions of ROC analysis are briefly recalled. The choice of an adequate (pseudo-)metric on the ROC space, a crucial point of the analysis, is also considered. The *smoothed bootstrap algorithm* is presented in Section 3, together with the theoretical results establishing its asymptotic accuracy as well as preliminary simulation results illustrating the impact of smoothing on the bootstrap performance. In Section 4, the gain in terms of convergence rate acquired by the smoothing step is thoroughly discussed. We refer to [1] for technical proofs.

## 2  Background

Here we briefly recall basic concepts of the bipartite ranking problem as well as key results related to the statistical estimation of ROC curves. We also set out the notations that shall be needed throughout the paper. Although the results contained in this paper can be formulated without referring to the bipartite ranking framework, in the purpose of motivating the present analysis we intentionally connected them to this major statistical learning problem, which has recently revitalized the interest for the problem of assessing the accuracy of empirical ROC curves, see [4].

### 2.1  Assumptions and notation

In the *bipartite ranking problem*, the problem is to order all the elements $X$ of a set $\mathcal{X}$ by degree of relevance, when relevancy may be observed through some binary indicator variable $Y$. Precisely, one has a system consisting of a binary random output $Y$, taking its values in $\{-1, 1\}$ say, and a random input $X$, taking its values in a (generally high-dimensional) feature space $\mathcal{X}$, which models some observation for predicting $Y$. The probabilistic model is the same as for standard binary classification but the prediction task is different. In the case of *information retrieval* for instance, the goal is to order all documents $x$ of the list $\mathcal{X}$ by degree of relevance for a particular request (rather than simply classifying them as relevant or not as in classification). This amounts to assigning to each document $x$ in $\mathcal{X}$ a *score* $s(x)$ indicating its degree of relevance for this specific query. The challenge is thus to build a *scoring function* $s : \mathcal{X} \to \mathbb{R}$ from sampling data, so as to rank the observations $x$ by increasing order of their score $s(x)$ as accurately as possible: the higher the score $s(X)$ is, the more likely one should observe $Y = +1$.

**True** ROC **curves.** A standard way of measuring the ranking performance consists of plotting the ROC curve, namely the graph of the mapping

$$\mathrm{ROC}_s : \alpha \in (0, 1) \mapsto 1 - (G_s \circ H_s^{-1})(1 - \alpha),$$

where $G_s$ (respectively $H_s$) denotes $s(X)$'s cdf conditioned on $Y = +1$ (resp. conditioned on $Y = -1$) and $F^{-1}(\alpha) = \inf\{x \in \mathbb{R}/ \ F(x) \geq \alpha\}$ the generalized inverse of any cdf $F$ on $\mathbb{R}$. It boils down to plotting the *true positive rate* versus the *false positive rate* when testing the assumption "$\mathcal{H}_0 : Y = -1$" based on the statistic $s(X)$. This functional performance measure induces a partial order on the set of scoring functions, according to which it may be shown, by standard Neyman-Pearson's arguments, that increasing transforms of the regression function $\eta(x) = \mathbb{P}(Y = +1 \mid X = x)$ are the optimal scoring functions (the test statistic $\eta(X)$ is *uniformly more powerful*, *i.e.* $\forall \alpha \in (0, 1), \mathrm{ROC}_\eta(\alpha) \geq \mathrm{ROC}_s(\alpha)$, for any scoring function $s(x)$).

**Empirical** ROC **curve estimates.** Practical learning strategies for selecting a good scoring function are based on training data $\mathcal{D}_n = \{(X_i, Y_i)\}_{1 \leq i \leq n}$ and should thus rely on accurate empirical estimates of the true ROC curves. Let $p = \mathbb{P}(Y = +1)$. For any scoring function candidate $s(X)$, an empirical counterpart of $\mathrm{ROC}_s$ is naturally obtained by computing

$$\forall \alpha \in (0, 1), \ \ \widehat{\mathrm{ROC}}_s(\alpha) = 1 - \widehat{G}_s \circ \widehat{H}_s^{-1}(1 - \alpha)$$

from empirical cdf estimates:

$$\widehat{G}_s(x) = \frac{1}{n_+} \sum_{i=1}^{n} \mathbb{I}_{\{Y_i = +1\}} K(x - s(X_i)) \text{ and } \widehat{H}_s(x) = \frac{1}{n_-} \sum_{i=1}^{n} \mathbb{I}_{\{Y_i = -1\}} K(x - s(X_i)),$$

where $n_+ = \sum_{i=1}^n \mathbb{I}\{Y_i = +1\} = n - n_-$ is the (random) number of positive instances among the sample (distributed as the binomial $Bin(n, p)$) and $K(u)$ denotes the step function $\mathbb{I}_{\{u \geq 0\}}$. In order to obtain smoothed versions $\widetilde{G}_s(x)$ and $\widetilde{F}_s(x)$ of the latter cdfs, a typical choice consists of picking instead a function $K(u)$ of the form $\int_{v \geq 0} \mathbf{K}_h(u - v)dv$, with $\mathbf{K}_h(u) = h^{-1}\mathbf{K}(h^{-1} \cdot u)$ where $\mathbf{K} \geq 0$ is a regularizing Parzen-Rosenblatt kernel (*i.e.* a bounded square integrable function such that $\int K(v)dv = 1$) and $h > 0$ is the smoothing bandwidth, see Remark 1 for a practical view of smoothing. Here and throughout, $\mathbb{I}_{\{\mathcal{A}\}}$ denotes the indicator function of any event $\mathcal{A}$.

**Metrics on the** ROC **space.** When it comes to measure closeness between curves in the ROC space, various metrics may be used, see [9]. Viewing the ROC space as a subset of the Skorohod's space $\mathbb{D}([0, 1])$ of *càd-làg* functions $f : [0, 1] \to \mathbb{R}$, the standard metric induced by the *sup norm* $||.||_\infty$ appears as a natural choice. As shall be seen below, asymptotic arguments for grounding the bootstrapping of the empirical ROC curve fluctuations, when measured in terms of the sup norm $||.||_\infty$, are rather straightforward. However, given the geometry of empirical ROC curves, this metric is not always convenient for our purpose and may produce very wide, and thus non informative confidence bands. For analyzing stepwise graphs, such as empirical ROC curves, we shall consider the closely related *pseudo-metric* defined as follows:

$$\forall (f_1, f_2) \in \mathbb{D}([0, 1])^2, \quad d_B(f_1, f_2) = \sup_{t \in [0,1]} d_B(f_1, f_2; t),$$

where $d_B(f_1, f_2; t) = \min\{|f_1(t) - f_2(t)|, \ |f_2^{-1} \circ f_1(t) - t|, \ |f_1^{-1} \circ f_2(t) - t|$. We clearly have $d_B(f_1, f_2) \leq ||f_1 - f_2||_\infty$. The major advantage of considering this pseudo-metric is that it provides a control on vertical and horizontal jumps of ROC curves both at the same time, treating both types of error in a symmetric fashion. Equipped with this pseudo-metric, two piecewise constant ROC curves may be close to each other, even if their jumps do not exactly match. This is clearly appropriate for describing the fluctuations of the empirical ROC curve (and the deviation between the latter and its bootstrap counterpart as well). This way, $d_B$ permits to construct builds bands of reasonable size, well adapted to the stepwise shape of empirical ROC curves, with better coverage probabilities. In this respect, the closely related *Hausdorff distance* (*i.e.* the distance between the graphs completed by linear segments at jump points) would also be a pertinent choice. However, providing a theoretical basis in the case of the Hausdorff distance is very challenging and will not be addressed in this paper, owing to space limitations.

As the goal pursued in the present paper is to build, in the ROC space viewed as a subspace of the Skorohod's space $\mathbb{D}([0, 1])$ equipped with a proper (pseudo-) metric, a confidence band for the ROC curve of a given diagnosis test statistic $s(X)$, we shall omit to index by $s$ the quantities considered and denote by $Z$ the r.v. $s(X)$ (and by $Z_i$, $1 \leq i \leq n$, the $s(X_i)$'s) for notational simplicity. Throughout the paper, we assume that $H(dx)$ and $G(dx)$ are continuous probability distributions, with densities $h(x)$ and $g(x)$ respectively. Eventually, denote by $\mathcal{P}$ the joint distribution of $(Z, Y)$ on $\mathbb{R} \times \{-1, +1\}$ and by $\mathcal{P}_n$ its empirical version based on the sample $\mathcal{D}_n = \{(Z_i, Y_i)\}_{1 \leq i \leq n}$. Equipped with the notations above, one may write $\mathcal{P}(dz, y) = p\mathbb{I}_{\{y=+1\}}G(dz) + (1 - p)\mathbb{I}_{\{y=-1\}}H(dz)$.

### 2.2 Asymptotic law - Gaussian approximation

In the situation described above, the next theorem establishes the strong consistency of the empirical ROC curve in *sup norm* and provides a *strong approximation* at the rate $1/\sqrt{n}$, up to logarithmic factors, for the *fluctuation process*:

$$r_n(\alpha) = \sqrt{n}(\widehat{\mathrm{ROC}}_n(\alpha) - \mathrm{ROC}(\alpha)), \quad \alpha \in [0, 1].$$

This (gaussian) approximation plays a crucial role in understanding the asymptotic behavior of the empirical ROC curve and of its bootstrap counterpart. The following assumptions are required.

   **H$_1$** The slope of the ROC curve is bounded: $\sup_{\alpha \in [0,1]}\{g(H^{-1}(\alpha))/h(H^{-1}(\alpha))\} < \infty$.

   **H$_2$** H is twice differentiable on $[0, 1]$. Furthermore, $\forall \alpha \in [0, 1]$, $h(\alpha) > 0$ and there exists $\gamma > 0$ such that $\sup_{\alpha \in [0,1]}\{\alpha(1 - \alpha) \cdot d\log(h \circ H^{-1}(\alpha))/d\alpha\} \leq \gamma < \infty$.

**Theorem. 1** (FUNCTIONAL LIMIT THEOREM) *Suppose that* **H$_1$** − **H$_2$** *are fulfilled. Then,*

*(i) the empirical* ROC *curve is strongly consistent:*

$$\sup_{\alpha \in [0,1]} |\widehat{\text{ROC}}_n(\alpha) - \text{ROC}(\alpha)| \to 0 \ a.s. \ as \ n \to \infty,$$

*(ii) there exist a sequence of two independent brownian bridges* $\{(B_1^{(n)}(\alpha), B_2^{(n)}(\alpha))\}_{\alpha \in [0,1]}$
*such that we almost surely have, uniformly over* $[0,1]$,

$$r_n(\alpha) = z^{(n)}(\alpha) + o\left((\log \log n)^{\rho_1(\gamma)}(\log n)^{\rho_2(\gamma)})/\sqrt{n}\right), \tag{1}$$

*where*

$$z^{(n)}(\alpha) = (1-p)^{-1/2} \frac{g(H^{-1}(1-\alpha))}{h(H^{-1}(1-\alpha))} B_1^{(n)}(\alpha) + p^{-1/2} B_2^{(n)}(\text{ROC}(\alpha))$$

*and*

$$\begin{cases} \rho_1(\gamma) &= 0, \ \rho_2(\gamma) = 1, \ \textit{if } \gamma < 1 \\ \rho_1(\gamma) &= 0, \ \rho_2(\gamma) = 2, \ \textit{if } \gamma = 1 \\ \rho_1(\gamma) &= \gamma, \ \rho_2(\gamma) = \gamma - 1 + \varepsilon, \ \varepsilon > 0, \ \textit{if } \gamma > 1 \end{cases}.$$

These results may be immediately derived from classical strong approximations for the empirical and quantile processes, see [5, 18]). Incidentally, we mention that the approximation rate is not always $\log^2(n)/\sqrt{n}$, contrarily to what is claimed in [18].

We point out that, owing to the presence of the term $(g/h)(H^{-1}(1-\alpha))$ in it, the gaussian approximant can hardly be used for constructing ROC confidence bands. To avoid explicit computation of density estimates, bootstrap confidence sets should be certainly preferred in practice.

## 3 Bootstrapping empirical ROC curves

Beyond consistency of the empirical curve in sup norm and the asymptotic normality of the fluctuation process, we now tackle the question of constructing confidence bands for the true ROC curve via the bootstrap approach introduced by [6], extending pointwise results established in [17]. The latter suggests to consider, as an estimate of the law of the fluctuation process $r_n = \{r_n(\alpha)\}_{\alpha \in [0,1]}$, the conditional law given $\mathcal{D}_n$ of the *bootstrapped fluctuation process*

$$r_n^* = \{\sqrt{n}(\text{ROC}^*(\alpha) - \widehat{\text{ROC}}(\alpha))\}_{\alpha \in [0,1]}, \tag{2}$$

where $ROC^*$ is the ROC curve corresponding to a sample $\mathcal{D}_n^* = \{(Z_i^*, Y_i^*)\}_{1 \le i \le n}$ of i.i.d. random pairs with a common distribution $\widetilde{\mathcal{P}}_n$ close to $\mathcal{P}_n$. We shall also consider

$$d_n^* = \sqrt{n} d_B(\text{ROC}^*, \widehat{\text{ROC}}), \tag{3}$$

whose random fluctuations, given $\mathcal{D}_n$, are expected to mimic those of $d_n = \sqrt{n} d_B(\widehat{\text{ROC}}, \text{ROC})$.

The difficulty is twofold. Firstly, the target of the bootstrap procedure is here a distribution on a *path space*, the ROC space being viewed as a subspace of $\mathbb{D}_n([0,1])$, equipped with either $||.||_\infty$ or else $d_B(.,.)$. Secondly, both $r_n$ and $d_n$ are functionals of the quantile process $\{\widehat{H}^{-1}(\alpha)\}_{\alpha \in [0,1]}$. It is well-known that the *naive bootstrap* (*i.e.* resampling from the raw empirical distribution) generally provides bad approximations of the distribution of empirical quantiles in practice: the rate of convergence for a given quantile is indeed of order $O_{\mathbb{P}}(n^{-1/4})$, see [7], whereas the rate of the gaussian approximation is $n^{-1/2}$. As shall be seen below, the same phenomenon may be naturally observed for ROC curves. In a similar fashion to what is generally recommended for empirical quantiles, we suggest to implement a *smoothed version* of the bootstrap algorithm in order to improve the approximation rate of $||r_n||_\infty$'s distribution, respectively of $d_n$'s distribution . In short, this boils down to resampling the data from a smoothed version of the empirical distribution $\mathcal{P}_n$.

### 3.1 The Algorithm

Here we describe the algorithm for building a confidence band at level $1 - \epsilon$ in the ROC space from sampling data $\mathcal{D}_n = \{(Z_i, Y_i); \ 1 \le i \le n\}$. Set $n_+ = \sum_{1 \le i \le n} \mathbb{I}_{\{Y_i=1\}} = n - n_-$. It is performed in four steps as follows.

Before turning to the theoretical properties of this algorithm and related numerical experiments, a few remarks are in order.

**Remark 1** *(MONTE-CARLO APPROXIMATION) From a computational angle, the true smoothed bootstrap distribution must be approximated in its turn, using a Monte-Carlo approximation scheme. A convenient way of doing this in practice, while reproducing theoretical advantages of smoothing, consists of drawing $B$ bootstrap samples, of size $n$, with replacement in the original data and then perturbating each drawn data by independent centered gaussian random variables of variance $h^2$ (this procedure is equivalent to drawing bootstrap data from a smooth estimate $\widetilde{\mathcal{P}}_n(dz, dy)$ computed using a gaussian kernel $K_h(u) = (2\pi h^2)^{-1/2}\exp(-u^2/(2h^2))$), see [22]. Regarding the choice of the number of bootstrap replications, picking $B = n$ does not modify the rate of convergence. However, choosing $B$ of magnitude comparable to $n$ so that $(1 + B)\epsilon$ is an integer may be more appropriate: the $\epsilon$-quantile of the approximate bootstrap distribution is the uniquely defined and this will not modify the rate of convergence neither, see [15].*

**Remark 2** *(ON TUNING PARAMETERS) The primary tuning parameters of the Algorithm are those related to the smoothing stage. When using a gaussian regularizing kernel, one should typically choose a bandwidth $h_n$ of order $n^{-1/5}$ in order to minimize the mean square error.*

**Remark 3** *(ON RECENTERING) From the asymptotic analysis viewpoint, it would be fairly equivalent to recenter by a smoothed version of the original empirical curve $\widetilde{ROC}(.) = 1 - \widetilde{G} \circ \widetilde{H}^{-1}(1 - .)$ in the computation of the bootstrap fluctuation process. However, numerically speaking, computing the sup norm of the estimate (2) is much more tractable, insofar as it solely requires to evaluate the distance between piecewise constant curves over the pooled set of jump points. It should also be noticed that smoothing the original curve, as proposed in [17], should be also avoided in practice, since it hides the jump locations, which constitute the essential part of the information.*

## 3.2 Asymptotic analysis

We now investigate the accuracy of the bootstrap estimate output by the Algorithm. The result stated in the next theorem extend those established in [17] in the *pointwise* framework. The functional nature of the approximation result below is essential, since it should be enhanced that, in most ranking applications, assessing the uncertainty about the whole estimated ROC curve, or some part of it at least, is what really matters. In the sequel, we assume that the kernel $K$ used in the smoothing step is "pyramidal" (*e.g.* gaussian or of the form $\mathbb{I}_{\{u \in [-1, +1]\}}$).

**Theorem. 2** *(ASYMPTOTIC ACCURACY) Suppose that the hypotheses of Theorem 1 are fulfilled. Assume further that smoothed versions of the cdf's $\widetilde{G}$ and $\widetilde{H}$ are computed at step 1 using a scaled kernel $\mathbf{K}_{h_n}(u)$ with $h_n \downarrow 0$ as $n \to \infty$ in a way that $nh_n^3 \to \infty$ and $nh_n^5 \log^2 n \to 0$. Then, the bootstrap distribution estimates output by the Algorithm are such that*

$$\sup_{t \in \mathbb{R}} |\mathbb{P}^*(||r_n^*||_\infty \leq t) - \mathbb{P}(||r_n||_\infty \leq t)| \ and \sup_{t \in \mathbb{R}} |\mathbb{P}^*(d_n^* \leq t) - \mathbb{P}(d_n \leq t)| \ are \ of \ order \ o_{\mathbb{P}}\left(\frac{\log(h_n^{-1})}{\sqrt{nh_n}}\right).$$

Hence, up to logarithmic factors, choosing $h_n \sim 1/(\log^{2+\eta} n^{1/5})$ with $\eta > 0$ yields an approximation error of order $n^{-2/5}$ for the bootstrap estimate. Although its rate is slower than the one of the gaussian approximation (1), the smoothed bootstrap method remains very appealing from a computational perspective, the construction of confidence bands from simulated brownian bridges being very difficult to implement in practice. As shall be seen below, the rate reached by the smoothed bootstrap distribution is nevertheless a great improvement, compared to the naive bootstrap approach (see the discussion below).

**Remark 4** *(BOOTSTRAPPING SUMMARY STATISTICS) From Theorem 1 above, asymptotic validity of the smooth bootstrap method for estimating the distribution of the fluctuations of a functional $\Phi(\widehat{\mathrm{ROC}})$ of the empirical ROC curve may be deduced, as soon as the function $\Phi$ defined on $\mathbb{D}([0,1])$ is sufficiently smooth (namely continuously Hadamard differentiable). For instance, it could be applied to summary statistics involving a specific piece of the ROC curve only in order to focus on the "best instances" [3], or more classically to the area under the ROC curve (AUC). However, in the latter case, due to the fact that this particular summary statistic is of the form of a U-statistic [2], the naive bootstrap rate is faster than the one we obtained here (of order $n^{-1}$).*

## 3.3 Simulation results

The striking advantage of the smoothed bootstrap is the improved rate of convergence of the resulting estimator. Furthermore, choosing $d_B$ for measuring the magnitude order of curve fluctuations has an even larger impact on the accuracy of the empirical bands. As an illustration of this theoretical result, we now display simulation results, emphasizing the gain acquired by smoothing and considering the pseudo-metric $d_B$.

We present confidence bands for a single trajectory and the estimation of the coverage probability of the bands for a simple *binormal model*:

$$Y_i = +1 \text{ if } \beta_0 + \beta_1 X + \varepsilon > 0, \text{ and } Y_i = -1 \text{ otherwise,}$$

where $\varepsilon$ and $X$ are independent standard normal r.v.'s. In this example, the scoring function $s(x)$ is the maximum likelihood estimator of the probit model on the training set. We choose here $\beta_0 = \beta_1 = 1$, $n = 1000$, $B = 999$ and $\gamma = 0.95$ for the targeted coverage probability. Coverage probabilities are obtained over 2000 replications of the procedure, using the package *ROCR* of statistical software *R*. As mentioned before, choosing $||.||_\infty$ yields very large bands with coverage probability close to 1! Though still large, bands based on the pseudo-metric $d_B$ are clearly much more informative (see Fig. 1). It should be noticed that the coverage improvement obtained by smoothing is clearer in the pontwise estimation setup (here $\alpha = 0.2$) but much more difficult to evidence for confidence bands.

Table 1: Empirical coverage probabilities for 95% empirical bands/intervals.

| METHOD | COVERAGE (%) |
|---|---|
| NAIVE BOOTSTRAP $||r_n||_\infty$ | 100 |
| SMOOTHED BOOTSTRAP $||r_n||_\infty$ | 100 |
| NAIVE BOOTSTRAP $d_n$ | 90.3 |
| SMOOTHED BOOTSTRAP $d_n$ | 93.1 |
| NAIVE BOOTSTRAP $r_n(0.2)$ | 89.7 |
| SMOOTHED BOOTSTRAP $r_n(0.2)$ | 92.5 |

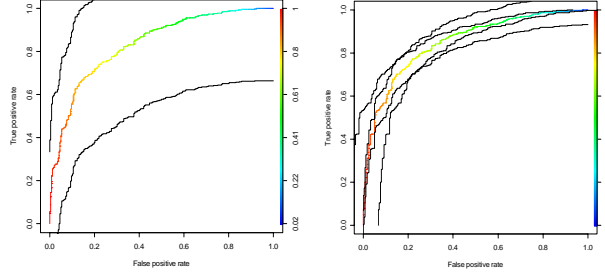

Figure 1 : $||.||_\infty$ confidence band  Figure 2 : $d_B$ confidence band

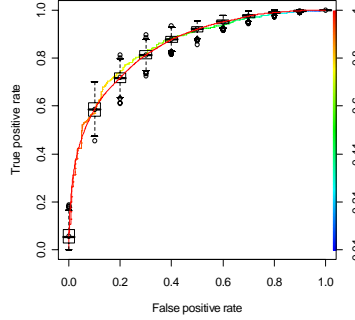

Figure 3: Ponctual smooth bootstrap confidence interval

Figure 1: ROC confidence bands.

## 4 Discussion

Let us now give an insight into the reason why the smoothed bootstrap procedure outperforms the bootstrap without smoothing. In most statistical problems where the nonparametric bootstrap is useful, there is no particular reason for implementing it from a smoothed version of the empirical df rather from the raw empirical distribution itself, see [22]. However, in the present case, smoothing affects the rate of convergence. Suppose indeed that the bootstrap process (2) is built by drawing from the raw cdf's $\widehat{G}$ and $\widehat{H}$ instead of their smoothed versions at step 2 of the Algorithm. Then, for any $\alpha \in ]0, 1[$, $\sup_{t \in \mathbb{R}} |\mathbb{P}^*(r_n^*(\alpha) \le t) - \mathbb{P}(r_n(\alpha) \le t)| = O_\mathbb{P}(n^{-1/4})$. Hence, the naive bootstrap rate induces an error of order $O(n^{-1/4})$ which cannot be improved, whereas it may be shown that the rate $n^{-2/5}$ is attained by the smoothed bootstrap (in a similar fashion to the functional setup), provided that the amount of smoothing is properly chosen. Heuristically, this is a consequence of the oscillation behavior of the deviation between the bootstrap quantile $H^{*-1}(1 - \alpha)$ and its expected value $\widehat{H}^{-1}(1 - \alpha)$ given the data $\mathcal{D}_n$, due to the fact that the step cdf $\widehat{H}$ is not regular around $\widehat{H}^{-1}(1 - \alpha)$: this corresponds to a jump with probability one.

**Higher-order accuracy.** A classical way of improving the pointwise approximation rate consists of bootstrapping a *standardized* version of the r.v. $r_n(\alpha)$. It is natural to consider, as standardization factor, the square root of an estimate of the asymptotic variance:

$$\sigma^2(\alpha) \quad = \quad var(z^{(n)}(\alpha)) = \frac{\alpha(1-\alpha)}{1-p} \frac{g(H^{-1}(1-\alpha))^2}{h(H^{-1}(1-\alpha))^2} + \frac{\text{ROC}(\alpha)(1 - \text{ROC}(\alpha))}{p}. \quad (4)$$

An estimate $\widehat{\sigma}_n^2$ of plug-in type could be considered, obtained by plugging $n_+/n$, $\widetilde{\text{ROC}}$ and smoothed density estimators $\tilde{h} = \widetilde{H}'$ and $\tilde{g} = \widetilde{G}'$ into (4) instead of their (unknown) theoretical counterparts. More interestingly, from a computational viewpoint, a bootstrap estimator of the variance could also be used. Following the argument used in [17] for a smoothed original estimate of the ROC curve, one may show that a smoothed bootstrap of the studentized statistic $r_n(\alpha)/\sigma_n(\alpha)$ yields a better pointwise rate of convergence than $1/\sqrt{n}$, the one of the gaussian approximation in the Central Limit Theorem. Precisely, for a given $\alpha \in ]0, 1[$, if the bandwidth used in the computation

of $\sigma_n^2(\alpha)$ is chosen of order $n^{-1/3}$, we have:

$$\sup_{t \in \mathbb{R}} \left| \mathbb{P}^* \left( \frac{r_n^*(\alpha)}{\sigma_n^*(\alpha)} \leq t \right) - \mathbb{P} \left( \frac{r_n(\alpha)}{\sigma_n(\alpha)} \leq t \right) \right| = O_{\mathbb{P}} \left( \frac{1}{n^{2/3}} \right), \tag{5}$$

denoting $\sigma_n^2(\alpha)$'s bootstrap counterpart by $\sigma_n^{*2}(\alpha)$. Notice that the bandwidth used in the standardization step (*i.e.* for estimating the variance) is not the same as the one used at the resampling stage of the procedure. This is a key point for achieving second-order accuracy. This time, the smoothed (studentized) bootstrap method widely outperforms the gaussian approach, when the matter is to build confidence intervals for the ordinate $\widehat{\text{ROC}}(\alpha)$ of a point of abciss $\alpha$ on the empirical ROC curve. However, it is not clear yet, whether this result remains true for confidence bands, when considering the whole ROC curve (this would actually require to establish an Edgeworth expansion for the supremum $||r_n/\widehat{\sigma}_n||_\infty$). This will be the scope of further research.

## References

[1] P. Bertail, S. Clémençon, and N. Vayatis. On constructing accurate confidence bands for ROC curves through smooth resampling, http://hal.archives-ouvertes.fr/hal-00335232/fr/. Technical report, 2008.

[2] S. Clémençon, G. Lugosi, and N. Vayatis. Ranking and scoring using empirical risk minimization. *Proceedings of COLT 2005, Eds P. Auer and R. Meir, LNAI 3559, Springer*, 2005.

[3] S. Clémençon and N. Vayatis. Ranking the best instances. *Journal of Machine Learning Research*, 5:197–227, 2007.

[4] W. Cohen, R. Schapire, and Y. Singer. Learning to order things. *Journal of Artificial Intelligence Research*, 10:243–270, 1999.

[5] M. Csorgo and P. Revesz. Strong approximations in probability and statistics. *Academic Press*, 1981.

[6] B. Efron. Bootstrap methods: another look at the jacknife. *Annals of Statistics*, 7:1–26, 1979.

[7] M. Falk and R. Reiss. Weak convergence of smoothed and nonsmoothed bootstrap quantile estimates. *Annals of Probability*, 17:362–371, 1989.

[8] T. Fawcett. ROC graphs: Notes and practical considerations for data mining researchers. *Technical Report HPL 2003-4)*, 5:197–227, 2003.

[9] P. Flach. The geometry of roc space: understanding machine learning metrics through roc isometrics. *In T. Fawcett and N. Mishra, editors, Proc. 20th International Conference on Machine Learning (ICML'03), AAAI Press*, 86:194–201, 2003.

[10] Y. Freund, R. Iyer, R. Schapire, and Y. Singer. An efficient boosting algorithm for combining preferences. *Journal of Machine Learning Research*, 4:933–969, 2003.

[11] P. Ghosal and J. Gu. Bayesian ROC curve estimation under binormality using a partial likelihood based on ranks. *Submitted for publication*, 2007.

[12] P. Ghosal and J. Gu. Strong approximations for resample quantile process and application to ROC methodology. *Submitted for publication*, 2007.

[13] A. Girling. ROC confidence bands: An empirical evaluation. *Journal of the Royal Statistical Society, Series B*, 62:367–382, 2000.

[14] D. Green and J. Swets. Signal detection theory and psychophysics. *Wiley, NY*, 1966.

[15] P. Hall. On the number of bootstrap simulations required to construct a confidence interval. *Annals of Statistics*, 14:1453–1462, 1986.

[16] P. Hall and R. Hyndman. Improved methods for bandwidth selection when estimating ROC curves. *Statistics and Probability Letters*, 64:181–189, 2003.

[17] P. Hall, R. Hyndman, and Y. Fan. Nonparametric confidence intervals for receiver operating characteristic curves. *Biometrika*, 91:743–750, 2004.

[18] F. Hsieh and B. Turnbull. Nonparametric and semi-parametric statistical estimation of the ROC curve. *The Annals of Statistics*, 24:25–40, 1996.

[19] S. Macskassy and F. Provost. Confidence bands for ROC curves: methods and an empirical study. *In Proceedings of the first Workshop on ROC Analysis in AI (ROCAI-2004) at ECAI-2004*, 2004.

[20] S. Macskassy, F. Provost, and S. Rosset. Bootstrapping the ROC curve: an empirical evaluation. *In Proceedings of ICML-2005 Workshop on ROC Analysis in Machine Learning (ROCML-2005)*, 2005.

[21] S. Macskassy, F. Provost, and S. Rosset. ROC confidence bands: An empirical evaluation. *In Proceedings of the 22nd International Conference on Machine Learning (ICML-2005)*, 2005.

[22] B. Silverman and G. Young. The bootstrap: to smooth or not to smooth? *Biometrika*, 74:469–479, 1987.

